# Learning to See Rotation and Dilation with a Hebb Rule

**Martin I. Sereno** and **Margaret E. Sereno**
Cognitive Science D-015
University of California, San Diego
La Jolla, CA 92093-0115

## Abstract

Previous work (M.I. Sereno, 1989; cf. M.E. Sereno, 1987) showed that a feedforward network with area V1-like input-layer units and a Hebb rule can develop area MT-like second layer units that solve the aperture problem for pattern motion. The present study extends this earlier work to more complex motions. Saito et al. (1986) showed that neurons with large receptive fields in macaque visual area MST are sensitive to different senses of rotation and dilation, irrespective of the receptive field location of the movement singularity. A network with an MT-like second layer was trained and tested on combinations of rotating, dilating, and translating patterns. Third-layer units learn to detect specific senses of rotation or dilation in a position-independent fashion, despite having position-dependent direction selectivity within their receptive fields.

## 1  INTRODUCTION

The visual systems of mammals and especially primates are capable of prodigious feats of movement, object, and scene recognition under noisy conditions--feats we would like to copy with artificial networks. We are just beginning to understand how biological networks are wired up during development and during learning in the adult. Even at this stage, however, it is clear that explicit error signals and the apparatus for propagating them backwards across layers are probably not involved. On the other hand, there is a growing body of evidence for connections whose strength can be modified (via NMDA channels) as functions of the correlation between pre- and post-synaptic activity. The present project was to try to learn to detect pattern rotation and dilation by example, using a simple Hebb

rule. By building up complex filters in stages using a simple, realistic learning rule, we reduce the complexity of what must be learned with more explicit supervision at higher levels.

## 1.1 ORIENTATION SELECTIVITY

Some of the connections responsible for the selectivity of cortical neurons to local stimulus features develop in the absence of patterned visual experience. For example, primary visual cortex (V1 or area 17) contains orientation-selective neurons at birth in several animals. Linsker (1986a,b) has shown that feedforward networks with gaussian topographic interlayer connections, linear summation, and simple hebb rules, develop orientation selective units in higher layers when trained on noise. In his linear system, weight updates for a layer can be written as a function of the two-point correlation characterizing the previous layer. Noise applied to the input layer causes the emergence of connections that generate gaussian correlations at the second layer. This in turn drives the development of more complex correlation functions in the third layer (e.g., difference-of-gaussians). Rotational symmetry is broken in higher layers with the emergence of Gabor-function-like connection patterns reminiscent of simple cells in the cortex.

## 1.2 PATTERN MOTION SELECTIVITY

The ability to see coherent motion fields develops late in primates. Human babies, for example, fail to see the transition from unstructured to structured motion--e.g., the transition between randomly moving dots and circular 2-D motion--for several months. The transition from horizontally moving dots with random y-axis velocities to dots with sinusoidal y-axis velocities (which gives the percept of a rotating 3-D cylinder) is seen even later (Spitz, Stiles-Davis, & Siegel, 1988). This suggests that the cortex requires many experiences of moving displays in order to learn how to recognize the various types of coherent texture motions.

However, orientation gradients, shape from shading, and pattern translation, dilation, and rotation cannot be detected with the kinds of filters that can be generated solely by noise. The correlations present in visual scenes are required in order for these higher level filters to arise.

## 1.3 NEUROPHYSIOLOGICAL MOTIVATION

Moving stimuli are processed in successive stages in primate visual cortical areas. The first cortical stage is layer $4C\alpha$ of V1, which receives its main ascending input from the magnocellular layers of the lateral geniculate nucleus. Layer $4C\alpha$ projects to layer 4B, which contains many tightly-tuned direction-selective neurons. These neurons, however, respond to moving contours as if these contours were moving perpendicular to their local orientation (Movshon et al., 1985).

Layer 4B neurons project directly and indirectly to area MT, where a subset of neurons show a relatively narrow peak in the direction tuning curve for a plaid that is lined up with the peak for a single grating. These neurons therefore solve the aperture problem for pattern translation presented to them by the local motion detectors in layer 4B of V1. MT neurons, however, appear to be largely blind to the sense of pattern rotation or dilation (Saito et al., 1986). Thus, there is a higher order 'aperture problem' that is solved by the neurons in the parts of areas MST and 7a that distinguish senses of pattern rotation and

dilation. The present model provides a rationale for how these stages might naturally arise in development.

## 2    RESULTS

In previous work (M.I. Sereno, 1989; cf. M.E. Sereno, 1987) a simple 2-layer feedforward architecture sufficed for an MT-like solution to the aperture problem for local translational motion. Units in the first layer were granted tuning curves like those in V1, layer 4B. Each first-layer unit responded to a particular range of directions and speeds of the component of movement perpendicular to a local contour. Second layer units developed MT-like receptive fields that solved the aperture problem for local pattern translation when trained on locally jiggled gratings rigidly moving in randomly chosen pattern directions.

### 2.1    NETWORK ARCHITECTURE

A similar architecture was used for second-to-third layer connections (see Fig. 1--a sample network with 5 directions and 3 speeds). As with Linsker, a new input layer was constructed from a canonical unit, suitably transformed. Thus, second-layer units were granted tuning curves resembling those found in MT (as well as those generated by first-to-second layer learning)--that is, they responded to the local *pattern* translation but were blind to particular senses of local rotation, dilation, and shear. There were 12 different local

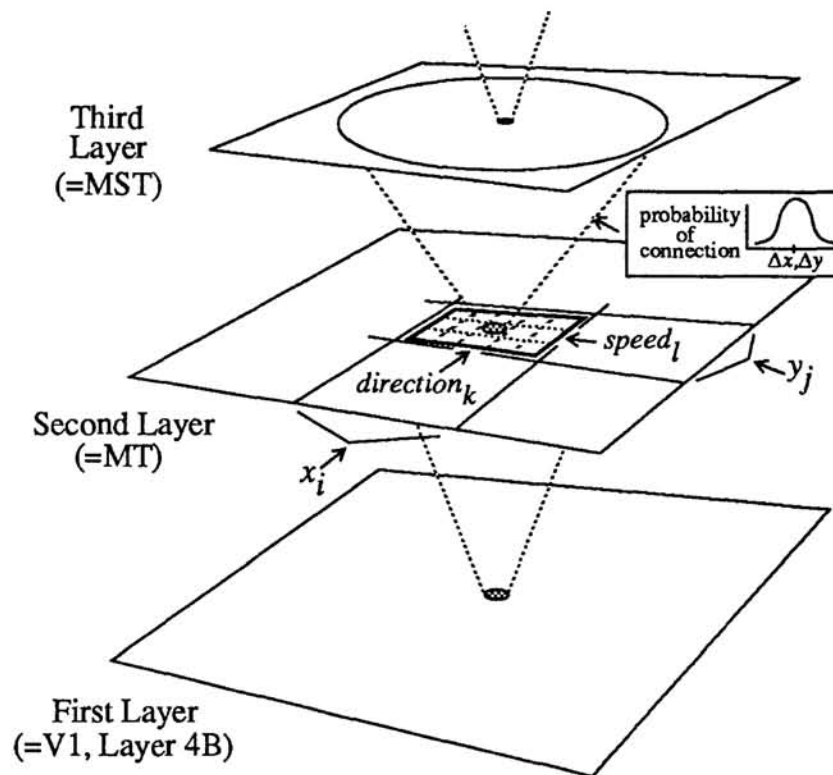

Figure 1:  Network Architecture

pattern directions and 4 different local pattern speeds at each x-y location (48 different units at each of 100 x-y points). Second-layer excitatory tuning curves were piecewise linear with half-height overlap for both direction and speed. Direction tuning was set to be 2-3 times as important as speed tuning in determining the activation of input units. Input units

generated untuned feedforward inhibition for off-directions and off-speeds. Total inhibition was adjusted to balance total excitation. The probability that a unit in the first layer connected to a unit in the second layer fell off as a gaussian centered on the retinotopically equivalent point in the second layer. Since receptive fields in areas MST and 7a are large, the interlayer divergence was increased relative to the divergence in the first-to-second layer connections. Third layer units received several thousand connections.

The network is similar to that of Linsker except that there is no activity-*independent* decay ($k_1$) for synaptic weights and no offset ($k_2$) for the correlation term. The activation, $out_j$, for each unit is a linear weighted sum of its inputs, $in_i$ scaled by $\alpha$, and clipped to maximum and minimum values:

$$out_j = \begin{cases} \alpha \sum_i in_i weight_{ij} \\ out_{max,min} \end{cases}$$

Weights are also clipped to maximum and minimum values. The change in each weight, $\Delta weight_{ij}$, is a simple fraction, $\delta$, of the product of the pre- and post-synaptic values:

$$\Delta weight_{ij} = \delta in_i out_j$$

The learning rate, $\delta$, was set so that about 1,000 patterns could be presented before most weights saturated. The stable second-layer weight patterns seen by Linsker (1986a) are reproduced by this model when it is trained on noise input. However, since it lacks $k_2$, it cannot generate center-surround weight structures given only gaussian correlations as input.

## 2.2   TRAINING PATTERNS

Second-to-third layer connections were trained with full or partial field rotations, dilations, and translations. Each stimulus consisted of a set of local pattern motions at each x-y point that were: 1) rotating clockwise or counterclockwise around, 2) dilating or contracting toward, or 3) translating through a randomly chosen location. The singularity was always within the input array. Both full and partial field rotations and dilations were effective training stimuli for generating rotation and dilation selectivity.

## 2.3   POSITION-INDEPENDENT TUNING CURVES

Post-training rotation and dilation tuning curves for different receptive-field locations were generated for many third-layer units using paradigms similar to those used on real neurons. The location of the motion singularity of the test stimulus was varied across layer two. Third-layer units often responded selectively to a particular sense of rotation or dilation at each visual field test location. A sizeable fraction of units (10-60%) responded in a position-independent way after unsupervised learning on rotating and dilating fields. Similar responses were found using both partial- and full-field test stimuli.

These units thus resemble the neurons in primate visual area MSTd (10-40% of the total there) recorded by Saito et al. (1986), Duffy and Wurtz (1990), and Andersen et al. (1990) that showed position-*independent* responses to rotations and dilations. Other third-layer units had position-*dependent* tuning--that is, they changed their selectivity for stimuli centered at different visual field locations, as, in fact, do a majority of actual MSTd neurons.

## 2.4  POSITION-DEPENDENT WEIGHT STRUCTURES

Given the position- independence of the selective response to rotations and/or dilations in some of the third-layer units, it was surprising to find that most such units had weight structures indicating that local direction sensitivity varied systematically across a unit's receptive field. Regions of maximum weights in direction-speed subspace tended to vary smoothly across x-y space such that opposite ends of the receptive field were sensitive to opposite directions. This picture obtained with full and medium-sized partial field training examples, breaking down only when the rotating and dilating training patterns were substantially smaller than the receptive fields of third-layer units. In the last case, smooth changes in direction selectivity across space were interrupted at intervals by discontinuities.

An essentially position-independent tuning curve is achieved because any off-center clockwise rotation that has its center within the receptive field of a unit selective for clockwise rotation will activate a much larger number of input units connected with large positive weights than will any off-center counterclockwise rotation (see Fig 2).

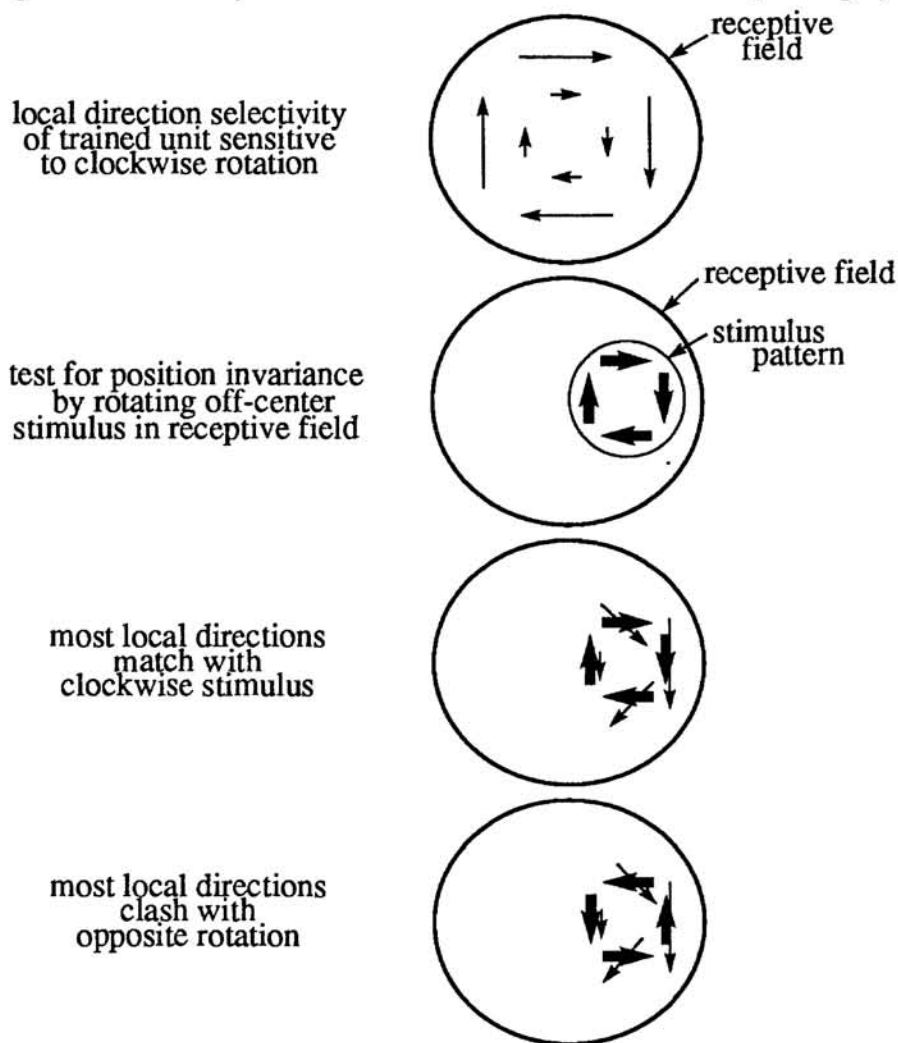

Figure 2:  Position-dependent weights and position-independent responses

Saito et al. (1986), Duffy & Wurtz (1990), and Andersen et al. (1990) have all suggested that true translationally-invariant detection of rotation and dilation sense must involve

several hierarchical processing stages and a complex connection pattern. The present results show that position-independent responses are exhibited by units with position-*dependent* local direction selectivity, as originally exhibited with small stimuli in area 7a by Motter and Mountcastle (1981).

## 2.5  WHY WEIGHTS ARE PERIODIC IN DIRECTION-SPEED SUBSPACE

For all training sets, the receptive fields of all units contained regions of all-max weights and all-min weights within the direction-speed subspace at each x-y point. For comparison, if the model is trained on uncorrelated direction noise (a different random local direction at each x-y point), third layer input weight structures still exhibit regions of all-max and all-min weights in the direction-speed subspace at each x-y point in the second layer. In contrast to weight structures generated by rigid motion, however, the location of these regions for a unit are not correlated across x-y space. These regions emerge at each x-y location because the overlap in the input unit tuning curves generates local two-point correlations in direction-speed subspace that are amplified by a hebb rule (Linsker, 1986a). This mechanism prevents more complex weight structures (like those envisaged by the neurophysiologists and those generated by backpropagation) from emerging. The two-point correlations *across x-y* space generated by jiggled gratings, or by the rotation and dilation training sets serve to align the all-max or all-min regions in the case of translation sensitivity, or generate smooth gradients in the case of sensitivity to rotation and dilation.

## 2.6  WHY MT DOES *NOT* LEARN TO DETECT ROTATION AND DILATION

Saito et al. (1986) demonstrated that MT neurons are not selective for particular senses of pattern rotation and dilation, but only for particular pattern translations (MT neurons will of course respond to a part of a large rotation or dilation that locally approximates the unit's translational directional tuning). MT neurons in the present model do not develop this selectivity even when trained on rotating and dilating stimuli because of the smaller divergence in the first layer (V1) to second layer (MT) connection. The local views of rotations and dilations seen by MT are apparently noise-like enough that any second order selectivity is averaged out. A larger (unrealistic) divergence allows a few units to solve the aperture problem and detect rotation and dilation in one step.

Training sets that contain many pure-translation stimuli along with the rotating and dilating stimuli fail to bring about the emergence of selectivity to senses of rotation and dilation (most units reliably detect only particular translations in this case). Satisfactory performance is achieved only if the translating stimuli are on average smaller than the rotating and dilating stimuli. This may point to a regularity in the poorly characterized stimulus set that the real visual system experiences, and perhaps in this case, has come to depend on for normal development.

## DISCUSSION

This exercise found a particularly simple solution to our problem that in retrospect should have been obvious from first principles. The present results suggest that this simple solution is also easily learned with simple Hebb rule. Two points warrant discussion.

First, this model achieves a reasonable degree of translational invariance in the detection of several simple kinds of pattern motion despite having weight structures that approximate a simple centered template. Such a solution to approximately translationally invariant

pattern detection may be applicable, and more importantly, practically learnable, for other more complex patterns, as long as the local features of interest vary reasonably smoothly and the pattern is not presented too far off-center. These constraints may characterize many foveated objects.

Second, given that the tuning curves for particular stimulus features often change in a continuous fashion as one moves across the cortex (e.g., orientation tuning, direction tuning), there is likely to be a pervasive tendency in the cortex for receptive fields in higher areas to be constructed from subunits that receive strong connections from nearby cells in the lower area.

## Acknowledgements

We thank Udo Wehmeier, Nigel Goddard, and David Zipser for help and discussions. Networks and displays were constructed on the Rochester Connectionist Simulator.

## References

Andersen, R., M. Graziano, and R. Snowden (1990) Translational invariance and attentional modulation of MST cells. *Soc. Neurosci., Abstr.* **16**:7.

Duffy, C.J. and R.H. Wurtz (1990) Organization of optic flow sensitive receptive fields in cortical area MST. *Soc. Neurosci., Abstr.* **16**:6.

Linsker, R. (1986a) From basic network principles to neural architecture: emergence of spatial-opponent cells. *Proc. Nat. Acad. Sci.* **83**, 7508-7512.

Linsker, R. (1986b) From basic network principles to neural architecture: emergence of orientation-selective cells. *Proc. Nat. Acad. Sci.* **83**, 8390-8394.

Motter, B.C. and V.B. Mountcastle (1981) The functional properties of the light-sensitive neurons of the posterior parietal cortex studied in waking monkeys: foveal sparing and opponent vector organization. *Jour. Neurosci.* **1**:3-26.

Movshon, J.A., E.H. Adelson, M.S. Gizzi, and W.T. Newsome (1985) Analysis of moving visual patterns. In C. Chagas, R. Gattass, and C. Gross (eds.), *Pattern Recognition Mechanisms.* Springer-Verlag, pp. 117-151.

Saito, H., M. Yukie, K. Tanaka, K. Hikosaka, Y. Fukada and E. Iwai (1986) Integration of direction signals of image motion in the superior temporal sulcus of the macaque monkey. *Jour. Neurosci.* **6**:145-157.

Sereno, M.E. (1987) Modeling stages of motion processing in neural networks. *Proceedings of the 9th Annual Cognitive Science Conference,* pp. 405-416.

Sereno, M.I. (1988) The visual system. In I.W.v. Seelen, U.M. Leinhos, & G. Shaw (eds.), *Organization of Neural Networks.* VCH, pp. 176-184.

Sereno, M.I. (1989) Learning the solution to the aperture problem for pattern motion with a hebb rule. In D.S. Touretzky (ed.), *Advances in Neural Information Processing Systems I.* Morgan Kaufmann Publishers, pp. 468-476.

R.V. Spitz, J. Stiles-Davis & R.M. Siegel. Infant perception of rotation from rigid structure-from-motion displays. *Soc. Neurosci., Abstr.* **14**, 1244 (1988).
